# The Electrotonic Transformation: a Tool for Relating Neuronal Form to Function

**Nicholas T. Carnevale**
Department of Psychology
Yale University
New Haven, CT 06520

**Kenneth Y. Tsai**
Department of Psychology
Yale University
New Haven, CT 06520

**Brenda J. Claiborne**
Division of Life Sciences
University of Texas
San Antonio, TX 79285

**Thomas H. Brown**
Department of Psychology
Yale University
New Haven, CT 06520

## Abstract

The spatial distribution and time course of electrical signals in neurons have important theoretical and practical consequences. Because it is difficult to infer how neuronal form affects electrical signaling, we have developed a quantitative yet intuitive approach to the analysis of electrotonus. This approach transforms the architecture of the cell from anatomical to electrotonic space, using the logarithm of voltage attenuation as the distance metric. We describe the theory behind this approach and illustrate its use.

## 1 INTRODUCTION

The fields of computational neuroscience and artificial neural nets have enjoyed a mutually beneficial exchange of ideas. This has been most evident at the network level, where concepts such as massive parallelism, lateral inhibition, and recurrent excitation have inspired both the analysis of brain circuits and the design of artificial neural net architectures.

Less attention has been given to how properties of the individual neurons or processing elements contribute to network function. Biological neurons and brain circuits have

been simultaneously subject to eons of evolutionary pressure. This suggests an essential interdependence between neuronal form and function, on the one hand, and the overall architecture and operation of biological neural nets, on the other. Therefore reverse-engineering the circuits of the brain appears likely to reveal design principles that rely upon neuronal properties. These principles may have maximum utility in the design of artificial neural nets that are constructed of processing elements with greater similarity to biological neurons than those which are used in contemporary designs.

Spatiotemporal extent is perhaps the most obvious difference between real neurons and processing elements. The processing element of most artificial neural nets is essentially a point in time and space. Its activation level is the instantaneous sum of its synaptic inputs. Of particular relevance to Hebbian learning rules, all synapses are exposed to the same activation level. These simplifications may insure analytical and implementational simplicity, but they deviate sharply from biological reality. Membrane potential, the biological counterpart of activation level, is neither instantaneous nor spatially uniform. Every cell has finite membrane capacitance, and all ionic currents are finite, so membrane potential must lag behind synaptic inputs. Furthermore, membrane capacitance and cytoplasmic resistance dictate that membrane potential will almost never be uniform throughout a living neuron embedded in the circuitry of the brain. The combination of ever-changing synaptic inputs with cellular anatomical and biophysical properties guarantees the existence of fluctuating electrical gradients.

Consider the task of building a massively parallel neural net from processing elements with such "nonideal" characteristics. Imagine moreover that the input surface of each processing element is an extensive, highly branched structure over which approximately 10,000 synaptic inputs are distributed. It might be tempting to try to minimize or work around the limitations imposed by device physics. However, a better strategy might be to exploit the computational consequences of these properties by making them part of the design, thereby turning these apparent weaknesses into strengths.

To facilitate an understanding of the spatiotemporal dynamics of electrical signaling in neurons, we have developed a new theoretical approach to linear electrotonus and a new way to make practical use of this theory. We present this method and illustrate its application to the analysis of synaptic interactions in hippocampal pyramidal cells.

## 2 THEORETICAL BACKGROUND

Our method draws upon and extends the results of two prior approaches: cable theory and two-port analysis.

### 2.1 CABLE THEORY

The modern use of cable theory in neuroscience began almost four decades ago with the work of Rall (1977). Much of the attraction of cable theory derives from the conceptual simplicity of the steady-state decay of voltage with distance along an infinite cylindrical cable: $V(x) = V_0 e^{-x/\lambda}$ where $x$ is physical distance and $\lambda$ is the length constant. This exponential relationship makes it useful to define the electrotonic distance $X$ as the

logarithm of the signal attenuation: $X = \ln V_0 / V(x)$. In an infinite cylindrical cable, electrotonic distance is directly proportional to physical distance: $X = x/\lambda$.

However, cable theory is difficult to apply to real neurons since dendritic trees are neither infinite nor cylindrical. Because of their anatomical complexity and irregular variations of branch diameter and length, attenuation in neurons is not an exponential function of distance. Even if a cell met the criteria that would allow its dendrites to be reduced to a finite equivalent cylinder (Rall 1977), voltage attenuation would not bear a simple exponential relationship to $X$ but instead would vary inversely with a hyperbolic function (Jack et al. 1983).

## 2.2 TWO-PORT THEORY

Because of the limitations and restrictions of cable theory, Carnevale and Johnston (1982) turned to two-port analysis. Among their conclusions, three are most relevant to this discussion.

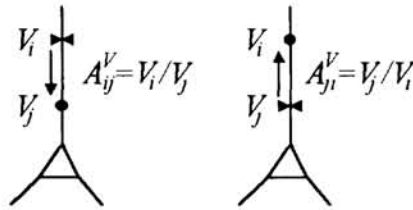

Figure 1: Attenuation is direction-dependent.

The first is that signal attenuation depends on the direction of signal propagation. Suppose that $i$ and $j$ are two points in a cell where $i$ is "upstream" from $j$ (voltage is spreading from $i$ to $j$), and define the voltage attenuation from $i$ to $j$: $A_{ij}^V = V_i / V_j$. Next suppose that the direction of signal propagation is reversed, so that $j$ is now upstream from $i$, and define the voltage attenuation $A_{ji}^V = V_j / V_i$. In general these two attenuations will not be equal: $A_{ij}^V \neq A_{ji}^V$

They also showed that voltage attenuation in one direction is identical to current attenuation in the opposite direction (Carnevale and Johnston 1982). Suppose current $I_i$ enters the cell at $i$, and the current that is captured by a voltage clamp at $j$ is $I_j$, and define the current attenuation $A_{ij}^I = I_i / I_j$. Because of the directional reciprocity between current and voltage attenuation, $A_{ij}^I = A_{ji}^V$. Similarly, if we interchange the current entry and voltage clamp sites, the current attenuation ratio would be $A_{ji}^I = A_{ij}^V$.

Finally, they found that charge and DC current attenuation in the same direction are identical (Carnevale and Johnston 1982). Therefore the spread of electrical signals between any two points is completely characterized by the voltage attenuations in both directions.

## 2.3  THE ELECTROTONIC TRANSFORMATION

The basic idea of the electrotonic transformation is to remap the cell from anatomical space into "electrotonic space," where the distance between points reflects the attenuation of an electrical signal spreading between them.  Because of the critical role of membrane potential in neuronal function, it is usually most appropriate to deal with voltage attenuations.

### 2.3.1  The Distance Metric

We use the logarithm of attenuation between points as the distance metric in electrotonic space: $L_{ij} = \ln A_{ij}$ (Brown et al. 1992, Zador et al. 1991).  To appreciate the utility of this definition, consider voltage spreading from point $i$ to point $j$, and suppose that $k$ is on the direct path between $i$ and $j$.  The voltage attenuations are $A_{ik}^V = V_i/V_k$, $A_{kj}^V = V_k/V_j$, and $A_{ij}^V = V_i/V_j = A_{ik}^V A_{kj}^V$.  This last equation and our definition of $L$ establish the additive property of electrotonic distance $L_{ij} = L_{ik} + L_{kj}$.  That is, electrotonic distances are additive over a path that has a consistent direction of signal propagation.  This justifies using the logarithm of attenuation as a metric for the electrical separation between points in a cell.

At this point several important facts should be noted.  First, unlike the electrotonic distance $X$ of classical cable theory, our new definition of electrotonic distance $L$ always bears a simple and direct logarithmic relationship to attenuation.  Second, because of membrane capacitance, attenuation increases with frequency.  Since both steady-state and transient signals are of interest, we evaluate attenuations at several different frequencies.  Third, attenuation is direction-dependent and usually asymmetric.  Therefore at every frequency of interest, each branch of the cell has two different representations in electrotonic space depending on the direction of signal flow.

### 2.3.2  Representing a Neuron in Electrotonic Space

Since attenuation depends on direction, it is necessary to construct transforms in pairs for each frequency of interest, one for signal spread away from a reference point ($V_{out}$) and the other for spread toward it ($V_{in}$).  The soma is often a good choice for the reference point, but any point in the cell could be used, and a different vantage point might be more appropriate for particular analyses.

The only difference between using one point $i$ as the reference instead of any other point $j$ is in the direction of signal propagation along the direct path between $i$ and $j$ (dashed arrows in Figure 2), where $V_{out}$ relative to $i$ is the same as $V_{in}$ relative to $j$ and vice versa.  The directions of signal flow and therefore the attenuations along all other branches of the cell are unchanged.  Thus the transforms relative to $i$ and $j$ differ only along the direct path $ij$, and once the $V_{out}$ and $V_{in}$ transforms have been created for one reference $i$, it is easy to assemble the transforms with respect to any other reference $j$.

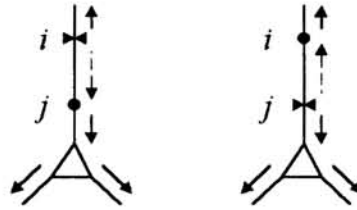

Figure 2: Effect of reference point location on direction of signal propagation.

We have found two graphical representations of the transform to be particularly useful. "Neuromorphic figures," in which the cell is redrawn so that the relative orientation of branches is preserved (Figures 3 and 4), can be readily compared to the original anatomy for quick, "big picture" insights regarding synaptic integration and interactions. For more quantitative analyses, it is helpful to plot electrotonic distance from the reference point as a function of anatomical distance (Tsai et al. 1993).

## 3 COMPUTATIONAL METHODS

The voltage attenuations along each segment of the cell are calculated from detailed, accurate morphometric data and the best available experimental estimates of the biophysical properties of membrane and cytoplasm. Any neural simulator like NEURON (Hines 1989) could be used to find the attenuations for the DC $V_{out}$ transform. The DC $V_{in}$ attenuations are more time consuming because a separate run must be performed for each of the dendritic terminations. However, the AC attenuations impose a severe computational burden on time-domain simulators because many cycles are required for the response to settle. For example, calculating the DC $V_{out}$ attenuations in a hippocampal pyramidal cell relative to the soma took only a few iterations on a SUN Sparc 10-40, but more than 20 hours were required for 40 Hz (Tsai et al. 1994). Finding the full set of attenuations for a $V_{in}$ transform at 40 Hz would have taken almost three months.

Therefore we designed an O(N) algorithm that achieves high computational efficiency by operating in the frequency domain and taking advantage of the branched architecture of the cell. In a series of recursive walks through the cell, the algorithm applies Kirchhoff's laws to compute the attenuations in each branch. The electrical characteristics of each segment of the cell are represented by an equivalent T circuit. Rather than "lump" the properties of cytoplasm and membrane into discrete resistances and capacitances, we determine the elements of these equivalent T circuits directly from complex impedance functions that we derived from the impulse response of a finite cable (Tsai et al. 1994). Since each segment is treated as a cable rather than an isopotential compartment, the resolution of the spatial grid does not affect accuracy, and there is no need to increase the resolution of the spatial grid in order to preserve accuracy as frequency increases. This is an important consideration for hippocampal neurons, which have long membrane time constants and begin to show increased attenuations at frequencies as low as 2 - 5 Hz (Tsai et al. 1994). It also allows us to treat a long unbranched neurite of nearly constant diameter as a single cylinder.

Thus runtimes scale linearly with the number of grid points, are independent of frequency, and we can even reduce the number of grid points if the diameters of adjacent

unbranched segments are similar enough. A benchmark of a program that uses our algorithm with NEURON showed a speedup of more than four orders of magnitude without sacrificing accuracy (2 seconds vs. 20 hours to calculate the $V_{out}$ attenuations at 40 Hz in a CA1 pyramidal neuron model with 2951 grid points) (Tsai et al. 1994).

## 4  RESULTS

### 4.1  DC TRANSFORMS OF A CA1 PYRAMIDAL CELL

Figure 3 shows a two-dimensional projection of the anatomy of a rat CA1 pyramidal neuron (cell 524, left) with neuromorphic renderings of its DC $V_{out}$ and $V_{in}$ transforms (middle and right) relative to the soma. The three-dimensional anatomical data were obtained from HRP-filled cells with a computer microscope system as described elsewhere (Rihn and Claiborne 1992, Claiborne 1992). The passive electrical properties used to compute the attenuations were $R_i = 200$ Ωcm, $C_m = 1$ μF/cm$^2$ (for nonzero frequencies, not shown here) and $R_m = 30$ kΩcm$^2$ (Spruston and Johnston 1992).

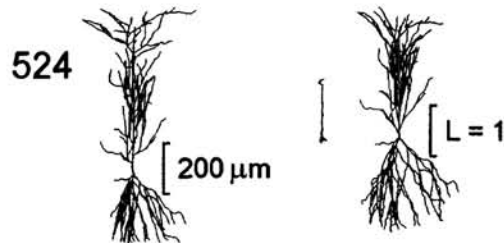

Figure 3:  CA1 pyramidal cell anatomy (cell 524, left) with neuromorphic renderings of $V_{out}$ (middle) and $V_{in}$ (right) transforms at DC.

The $V_{out}$ transform is very compact, indicating that voltage propagates from the soma into the dendrites with relatively little attenuation. The basilar dendrites and the terminal branches of the primary apical dendrite are almost invisible, since they are nearly isopotential along their lengths. Despite the fact that the primary apical dendrite has a larger diameter than any of its daughter branches, most of the voltage drop for somatofugal signaling is in the primary apical. Therefore it accounts for almost all of the electrotonic length of the cell in the $V_{out}$ transform.

The $V_{in}$ transform is far more extensive, but most of the electrotonic length of the cell is now in the basilar and terminal apical branches. This reflects the loading effect of downstream membrane on these thin dendritic branches.

### 4.2  SYNAPTIC INTERACTIONS

The transform can also give clues to possible effects of electrotonic architecture on voltage-dependent forms of associative synaptic plasticity and other kinds of synaptic interactions. Suppose the cell of Figure 3 receives a weak or "student" synaptic input

located 400 μm from the soma on the primary apical dendrite, and a strong or "teacher" input is situated 300 μm from the soma on the same dendrite.

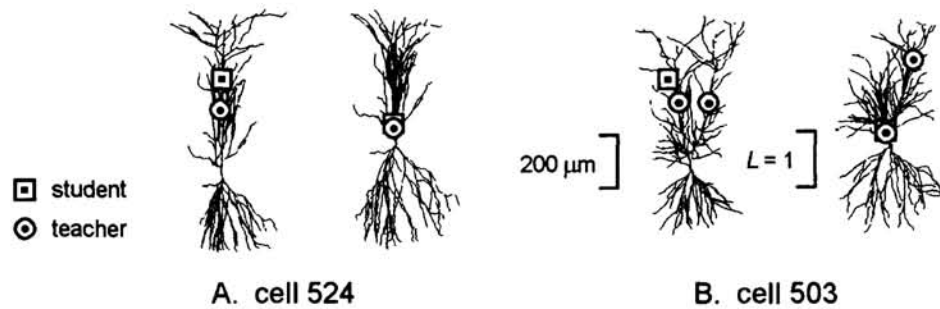

□ student
◉ teacher

A. cell 524          B. cell 503

Figure 4: Analysis of synaptic interactions.

The anatomical arrangement is depicted on the left in Figure 4A ("student" = square, "teacher" = circle). The $V_{in}$ transform with respect to the student (right figure of this pair) shows that voltage spreads from the teacher to the student synapse with little attenuation, which would favor voltage-dependent associative interactions.

Figure 4B shows a different CA1 pyramidal cell in which the apical dendrite bifurcates shortly after arising from the soma. Two teacher synapses are indicated, one on the same branch as the student and the other on the opposite branch. The $V_{in}$ transform with respect to the student (right figure of this pair) shows clearly that the teacher synapse on the same branch is closely coupled to the student, but the other is electrically much more remote and less likely to influence the student synapse.

## 5. SUMMARY

The electrotonic transformation is based on a logical, internally consistent conceptual approach to understanding the propagation of electrical signals in neurons. In this paper we described two methods for graphically presenting the results of the transformation: neuromorphic rendering, and plots of electrotonic distance vs. anatomical distance. Using neuromorphic renderings, we illustrated the electrotonic properties of a previously unreported hippocampal CA1 pyramidal neuron as viewed from the soma (cell 524, Figure 3). We also extended the use of the transformation to the study of associative interactions between "teacher" and "student" synapses by analyzing this cell from the viewpoint of a "student" synapse located in the apical dendrites, contrasting this result with a different cell that had a bifurcated primary apical dendrite (cell 503, Figure 4). This demonstrates the versatility of the electrotonic transformation, and shows how it can convey the electrical signaling properties of neurons in ways that are quickly and easily comprehended.

This understanding is important for several reasons. First, electrotonus affects the integration and interaction of synaptic inputs, regulates voltage-dependent mechanisms of synaptic plasticity, and influences the interpretation of intracellular recordings. In addition, phylogeny, development, aging, and response to injury and disease are all accompanied by alterations of neuronal morphology, some subtle and some profound.

The significance of these changes for brain function becomes clear only if their effects on neuronal signaling are reckoned. Finally, there is good reason to expect that neuronal electrotonus is highly relevant to the design of artificial neural networks.

## Acknowledgments

We thank R.B. Gonzales and M.P. O'Boyle for their contributions to the morphometric analysis, and Z.F. Mainen for assisting in the initial development of graphical rendering. This work was supported in part by ONR, ARPA, and the Yale Center for Theoretical and Applied Neuroscience (CTAN).

## References

Brown, T.H., Zador, A.M., Mainen, Z.F. and Claiborne, B.J. Hebbian computations in hippocampal dendrites and spines. In: *Single Neuron Computation*, eds. McKenna, T., Davis, J. and Zornetzer, S.F., New York, Academic Press, 1992, pp. 81-116.

Carnevale, N.T. and Johnston, D.. Electrophysiological characterization of remote chemical synapses. *J. Neurophysiol. 47*:606-621, 1982.

Claiborne, B.J. The use of computers for the quantitative, three-dimensional analysis of dendritic trees. In: *Methods in Neuroscience. Vol. 10: Computers and Computation in the Neurosciences*, ed. Conn, P.M., New York, Academic Press, 1992, pp. 315-330.

Hines, M. A program for simulation of nerve equations with branching geometries. *Internat. J. Bio-Med. Computat.* 24:55-68, 1989.

Rall, W.. Core conductor theory and cable properties of neurons. In: *Handbook of Physiology, The Nervous System*, ed. Kandel, E.R., Bethesda, MD, Am. Physiol. Soc., 1977, pp.39-98.

Rihn, L.L. and Claiborne, B.J. Dendritic growth and regression in rat dentate granule cells during late postnatal development. *Brain Res. Dev.* 54(1):115-24, 1990.

Spruston, N. and Johnston, D. Perforated patch-clamp analysis of the passive membrane properties of three classes of hippocampal neurons. *J. Neurophysiol.* 67:508-529, 1992.

Tsai, K.Y., Carnevale, N.T., Claiborne, B.J. and Brown, T.H. Morphoelectrotonic transforms in three classes of rat hippocampal neurons. *Soc. Neurosci. Abst.* 19:1522, 1993.

Tsai, K.Y., Carnevale, N.T., Claiborne, B.J. and Brown, T.H. Efficient mapping from neuroanatomical to electrotonic space. *Network* 5:21-46, 1994.

Zador, A.M., Claiborne, B.J. and Brown, T.H. Electrotonic transforms of hippocampal neurons. *Soc. Neurosci. Abst.* 17:1515, 1991.